# Distributed Information Regularization on Graphs

**Adrian Corduneanu**
CSAIL MIT
Cambridge, MA 02139
adrianc@csail.mit.edu

**Tommi Jaakkola**
CSAIL MIT
Cambridge, MA 02139
tommi@csail.mit.edu

## Abstract

We provide a principle for semi-supervised learning based on optimizing
the rate of communicating labels for unlabeled points with side informa-
tion. The side information is expressed in terms of identities of sets of
points or regions with the purpose of biasing the labels in each region
to be the same. The resulting regularization objective is convex, has a
unique solution, and the solution can be found with a pair of local prop-
agation operations on graphs induced by the regions. We analyze the
properties of the algorithm and demonstrate its performance on docu-
ment classification tasks.

## 1 Introduction

A number of approaches and algorithms have been proposed for semi-supervised learning
including parametric models [1], random field/walk models [2, 3], or discriminative (kernel
based) approaches [4]. The basic intuition underlying these methods is that the labels
should not change within clusters of points, where the definition of a cluster may vary from
one method to another.

We provide here an alternative information theoretic criterion and associated algorithms
for solving semi-supervised learning problems. Our formulation, an extension of [5, 6],
is based on the idea of minimizing the number of bits required to communicate labels for
unlabeled points, and involves no parametric assumptions. The communication scheme
inherent to the approach is defined in terms of regions, weighted sets of points, that are
shared between the sender and the receiver. The regions are important in capturing the
topology over the points to be labeled, and, through the communication criterion, bias the
labels to be the same within each region.

We start by defining the communication game and the associated regularization problem,
analyze properties of the regularizer, derive distributed algorithms for finding the unique
solution to the regularization problem, and demonstrate the method on a document classi-
fication task.

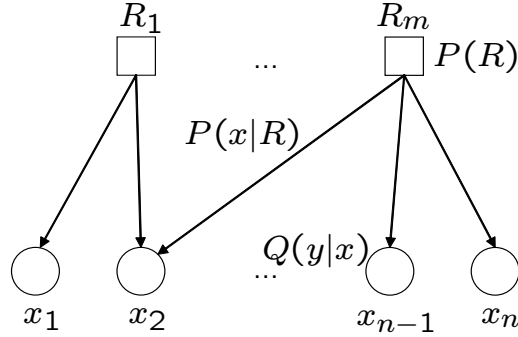

Figure 1: The topology imposed by the set of regions (squares) on unlabeled points (circles)

## 2  The communication problem

Let $\mathcal{S} = \{x_1, \ldots, x_n\}$ be the set of unlabeled points and $\mathcal{Y}$ the set of possible labels. We assume that target labels are available only for a small subset $\mathcal{S}_l \subset \mathcal{S}$ of the unlabeled points. The objective here is to find a conditional distribution $Q(y|x)$ over the labels at each unlabeled point $x \in \mathcal{S}$. The estimation is made possible by a regularization criterion over the conditionals which we define here through a communication problem. The communication scheme relies on a set of regions $\mathcal{R} = \{R_1, \ldots, R_m\}$, where each region $R \in \mathcal{R}$ is a subset of the unlabeled points $\mathcal{S}$ (cf. Figure 1). The weights of points within each region are expressed in terms of a conditional distribution $P(x|R)$, $\sum_{x \in R} P(x|R) = 1$, and each region has an *a priori* probability $P(R)$. We require only that $\sum_{R \in \mathcal{R}} P(x|R)P(R) = 1/n$ for all $x \in \mathcal{S}$. (Note: in our overloaded notation "$R$" stands both for the set of points and its identity as a set).

The regions and the membership probabilities are set in an application specific manner. For example, in a document classification setting we might define regions as sets of documents containing each word. The probabilities $P(R)$ and $P(x|R)$ could be subsequently derived from a word frequency representation of documents: if $f(w|x)$ is the frequency of word $w$ in document $x$, then for each pair of $w$ and the corresponding region $R$ we can set $P(R) = \sum_{x \in \mathcal{S}} f(w|x)/n$ and $P(x|R) = f(w|x)/(nP(R))$.

For any fixed conditionals $\{Q(y|x)\}$ we define the communication problem as follows. The sender selects a region $R \in \mathcal{R}$ with probability $P(R)$ and a point within the region according to $P(x|R)$. Since $\sum_{R \in \mathcal{R}} P(x|R)P(R) = 1/n$, each point $x$ is overall equally likely to be selected. The label $y$ is sampled from $Q(y|x)$ and communicated to the receiver optimally using a coding scheme tailored to the region $R$ (based on knowing $P(x|R)$ and $Q(y|x)$, $x \in R$). The receiver has access to $x$, $R$, and the region specific coding scheme to reproduce $y$. The rate of information needed to be sent to the receiver in this scheme is given by

$$J_c(Q; \mathcal{R}) = \sum_{R \in \mathcal{R}} P(R) I_R(x; y) = \sum_{R \in \mathcal{R}} P(R) \sum_{x \in R} \sum_{y \in \mathcal{Y}} P(x|R) Q(y|x) \log \frac{Q(y|x)}{Q(y|R)} \quad (1)$$

where $Q(y|R) = \sum_{x \in R} P(x|R)Q(y|x)$ is the overall probability of $y$ within the region.

## 3 The regularization problem

We use $J_c(Q; \mathcal{R})$ to regularize the conditionals. This regularizer biases the conditional distributions to be constant within each region so as to minimize the communication cost $I_R(x; y)$. Put another way, by introducing a region $R$ we bias the points in the region to be labeled the same. By adding the cost of encoding the few available labeled points, expressed here in terms of the empirical distribution $\hat{P}(y, x)$ whose support lies in $\mathcal{S}_l$, the overall regularization criterion is given by

$$J(Q; \lambda) = - \sum_{x \in \mathcal{S}_l} \sum_{y \in \mathcal{Y}} \hat{P}(y, x) \log Q(y|x) + \lambda J_c(Q; \mathcal{R}) \tag{2}$$

where $\lambda > 0$ is a regularization parameter. The following lemma guarantees that the solution is always unique:

**Lemma 1** $J(Q; \lambda)$ *for $\lambda > 0$ is a strictly convex function of the conditionals $\{Q(y|x)\}$ provided that 1) each point $x \in \mathcal{S}$ belongs to at least one region containing at least two points, and 2) the membership probabilities $P(x|R)$ and $P(R)$ are all non-zero.*

The proof follows immediately from the strict convexity of mutual information [7] and the fact that the two conditions guarantee that each $Q(y|x)$ appears non-trivially in at least one mutual information term.

## 4 Regularizer and the number of labelings

We consider here a simple setting where the labels are hard and binary, $Q(y|x) \in \{0, 1\}$, and seek to bound the number of possible binary labelings consistent with a cap on the regularizer.

We assume for simplicity that points in a region have uniform weights $P(x|R)$. Let $N(I)$ be the number of labelings of $\mathcal{S}$ consistent with an upper bound $I$ on the regularizer $J_c(Q, \mathcal{R})$. The goal is to show that $N(I)$ is significantly less than $2^n$ and $N(I) \to 2$ as $I \to 0$.

**Theorem 2** $\log_2 N(I) \le C(I) + I \cdot n \cdot t(\mathcal{R})/ \min_R P(R)$*, where $C(I) \to 1$ as $I \to 0$, and $t(\mathcal{R})$ is a property of $\mathcal{R}$.*

**Proof** Let $f(R)$ be the fraction of positive samples in region $R$. Because the labels are binary $I_R(x; y)$ is given by $H(f(R))$, where $H$ is the entropy. If $\sum_R P(R)H(f(R)) \le I$ then certainly $H(f(R)) \le I/P(R)$. Since the binary entropy is concave and symmetric w.r.t. 0.5, this is equivalent to $f(R) \le g_R(I)$ or $f(R) >= 1 - g_R(I)$, where $g_R(I)$ is the inverse of $H$ at $I/P(R)$. We say that a region is mainly negative if the former condition holds, or mainly positive if the latter.

If two regions $R_1$ and $R_2$ overlap by a large amount, they must be mainly positive or mainly negative together. Specifically this is the case if $|R_1 \cap R_2| > g_{R_1}(I)|R_1| + g_{R_2}(I)|R_2|$ Consider a graph with vertices the regions, and edges whenever the above condition holds. Then regions in a connected component must be all mainly positive or mainly negative together. Let $C(I)$ be the number of connected components in this graph, and note that $C(I) \to 1$ as $I \to 0$.

We upper bound the number of labelings of the points spanned by a given connected component $\mathcal{C}$, and subsequently combine the bounds. Consider the case in which all regions in $\mathcal{C}$ are mainly negative. For any subset $\mathcal{C}'$ of $\mathcal{C}$ that still covers all the points spanned by $\mathcal{C}$,

$$f(\mathcal{C}) \le \frac{1}{|\mathcal{C}|} \sum_{R \in \mathcal{C}'} g_I(R)|R| \le \max_R g_I(R) \cdot \frac{\sum_{R \in \mathcal{C}'} |R|}{|\mathcal{C}'|} \tag{3}$$

Thus $f(\mathcal{C}) \leq t(\mathcal{C}) \max_R g_I(R)$ where $t(\mathcal{C}) = \min_{\mathcal{C}' \in \mathcal{C}, \ \mathcal{C}' \text{ cover}} \frac{\sum_{\mathcal{R} \in \mathcal{C}'} |\mathcal{R}|}{|\mathcal{C}'|}$ is the minimum average number of times a point in $\mathcal{C}$ is necessarily covered.

There at most $2^{nf(R) \log_2(2/f(R))}$ labelings of a set of points of which at most $nf(R)$ are positive. [1]. Thus the number of feasible labelings of the connected component $\mathcal{C}$ is upper bounded by $2^{1 + nt(\mathcal{C}) \max_R g_I(R) \log_2(2/(t(\mathcal{C}) \max_R g_I(R)))}$ where 1 is because $\mathcal{C}$ can be either mainly positive or mainly negative. By cumulating the bounds over all connected components and upper bounding the entropy-like term with $I/P(R)$ we achieve the stated result. $\square$

Note that $t(\mathcal{R})$, the average number of times a point is covered by a minimal subcovering of $\mathcal{R}$ normally does not scale with $|\mathcal{R}|$ and is a covering dependent constant.

## 5 Distributed propagation algorithm

We introduce here a local propagation algorithm for minimizing $J(Q; \lambda)$ that is both easy to implement and provably convergent. The algorithm can be seen as a variant of the Blahut-Arimoto algorithm in rate-distortion theory [8], adapted to the more structured context here. We begin by rewriting each mutual information term $I_R(x; y)$ in the criterion

$$I_R(x; y) = \sum_{x \in R} \sum_{y \in \mathcal{Y}} P(x|R) Q(y|x) \log \frac{Q(y|x)}{Q(y|R)} \tag{4}$$

$$= \min_{Q_R(\cdot)} \sum_{x \in R} \sum_{y \in \mathcal{Y}} P(x|R) Q(y|x) \log \frac{Q(y|x)}{Q_R(y)} \tag{5}$$

where the variational distribution $Q_R(y)$ can be chosen independently from $Q(y|x)$ but the unique minimum is attained when $Q_R(y) = Q(y|R) = \sum_{x \in R} P(x|R) Q(y|x)$. We can extend the regularizer over both $\{Q(y|x)\}$ and $\{Q_R(y)\}$ by defining

$$J_c(Q, Q_R; \mathcal{R}) = \sum_{R \in \mathcal{R}} P(R) \sum_{x \in R} \sum_{y \in \mathcal{Y}} P(x|R) Q(y|x) \log \frac{Q(y|x)}{Q_R(y)} \tag{6}$$

so that $J_c(Q; \mathcal{R}) = \min_{\{Q_R(\cdot), R \in \mathcal{R}\}} J_c(Q, Q_R; \mathcal{R})$ recovers the original regularizer.

The local propagation algorithm follows from optimizing each $Q(y|x)$ based on fixed $\{Q_R(y)\}$ and subsequently finding each $Q_R(y)$ given fixed $\{Q(y|x)\}$. We omit the straightforward derivation and provide only the resulting algorithm: for all points $x \in S \cap S_l$ (not labeled) and for all regions $R \in \mathcal{R}$ we perform the following complementary averaging updates

$$Q(y|x) \leftarrow \frac{1}{Z_x} \exp\left( \sum_{R: x \in R} [nP(R)P(x|R)] \log Q_R(y) \right) \tag{7}$$

$$Q_R(y) \leftarrow \sum_{x \in R} P(x|R) Q(y|x) \tag{8}$$

where $Z_x$ is a normalization constant. In other words, $Q(y|x)$ is obtained by taking a weighted geometric average of the distributions associated with the regions, whereas $Q_R(y)$ is (as before) a weighted arithmetic average of the conditionals within each region. In terms of the document classification example discussed earlier, the weight $[nP(R)P(x|R)]$ appearing in the geometric average reduces to $f(w|x)$, the frequency of word $w$ identified with region $R$ in document $x$.

Updating $Q(y|x)$ for each labeled point $x \in \mathcal{S}_l$ involves minimizing

$$
\sum_{y \in \mathcal{Y}} \hat{P}(y, x) \log Q(y|x) - \frac{\lambda}{n} H(Q(\cdot|x)) -
$$
$$
- \lambda \sum_{y \in \mathcal{Y}} Q(y|x) \left( \sum_{R:x \in R} P(R) P(x|R) \log Q_R(y) \right) \tag{9}
$$

where $H(Q(\cdot|x))$ is the Shannon entropy of the conditional. While the objective is strictly convex, the solution cannot be written in closed form and have to be found iteratively (e.g., via Newton-Raphson or simple bracketing when the labels are binary). A much simpler update $Q(y|x) = \delta(y, \hat{y}_x)$, where $\hat{y}_x$ is the observed label for $x$, may suffice in practice. This update results from taking the limit of small $\lambda$ and approximates the iterative solution.

## 6 Extensions

### 6.1 Structured labels and generalized propagation steps

Here we extend the regularization framework to the case where the labels represent more structured annotations of objects. Let $y$ be a vector of elementary labels $y = [y_1, \ldots, y_k]'$ associated with a single object $x$. We assume that the distribution $Q(y|x) = Q(y_1, \ldots, y_k|x)$, for any $x$, can be represented as a tree structured graphical model, where the structure is the same for all $x \in \mathcal{S}$. The model is appropriate, e.g., in the context of assigning topics to documents. While the regularization principle applies directly if we leave $Q(y|x)$ unconstrained, the calculations would be potentially infeasible due to the number of elementary labels involved, and inefficient as we would not explicitly make use of the assumed structure. Consequently, we seek to extend the regularization framework to handle distributions of the form

$$
Q_{\mathcal{T}}(y|x) = \prod_{i=1}^{k} Q_i(y_i|x) \prod_{(i,j) \in \mathcal{T}} \frac{Q_{ij}(y_i, y_j|x)}{Q_i(y_i|x) Q_j(y_j|x)} \tag{10}
$$

where $\mathcal{T}$ defines the edge set of the tree. The regularization problem will be formulated over $\{Q_i(y_i|x), Q_{ij}(y_i, y_j|x)\}$ rather than unconstrained $Q(y|x)$.

The difficulty in this case arises from the fact that the arithmetic average (mixing) in eq (8) is not structure preserving (tree structured models are not mean flat). We can, however, also constrain $Q_R(y)$ to factor according to the same tree structure. By restricting the class of variational distributions $Q_R(y)$ that we consider, we necessarily obtain an *upper bound* on the original information criterion. If we minimize this upper bound with respect to $\{Q_R(y)\}$, under the factorization constraint

$$
Q_{R,\mathcal{T}}(y) = \prod_{i=1}^{k} Q_{R,i}(y_i) \prod_{(i,j) \in \mathcal{T}} \frac{Q_{R,ij}(y_i, y_j)}{Q_{R,i}(y_i|x) Q_{R,j}(y_j)}, \tag{11}
$$

given fixed $\{Q_{\mathcal{T}}(y|x)\}$, we can replace eq (8) with simple "moment matching" updates

$$
Q_{R,ij}(y_i, y_j) \leftarrow \sum_{x \in R} P(x|R) Q_{ij}(y_i, y_j|x) \tag{12}
$$

The geometric update of $Q(y|x)$ in eq (7) is structure preserving in the sense that if $Q_{R,\mathcal{T}}(y)$, $R \in \mathcal{R}$ share the same tree structure, then so will the resulting conditional. The new updates will result in a monotonically decreasing bound on the original criterion.

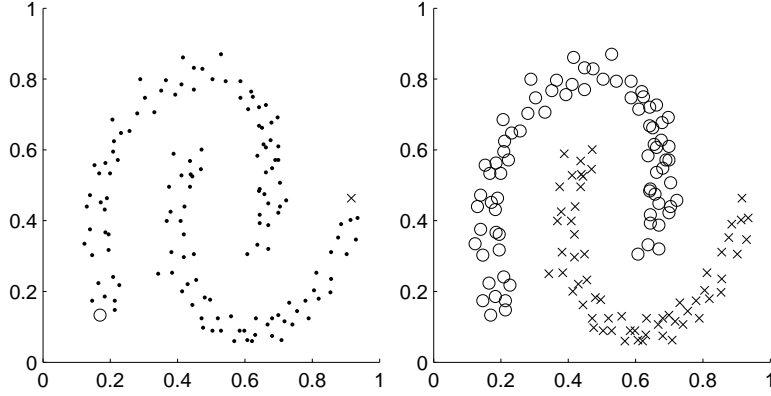

Figure 2: Clusters correctly separated by information regularization given one label from each class

## 6.2  Complementary sets of regions

In many cases the points to be labeled may have alternative feature representations, each leading to a different set of natural regions $\mathcal{R}^{(k)}$. For example, in web page classification both the content of the page, and the type of documents that link to that page should be correlated with its topic. The relationship between these heterogeneous features may be complex, with some features more relevant to the classification task than others.

Let $J_c(Q; \mathcal{R}^{(k)})$ denote the regularizer from the $k^{th}$ feature representation. Since the regularizers are on a common scale we can combine them linearly:

$$J_c(Q; \mathcal{K}, \alpha) = \sum_{k=1}^{K} \alpha_k J_c(Q; \mathcal{R}^{(k)}) = \sum_{k=1}^{K} \sum_{R \in \mathcal{R}^{(k)}} \alpha_k P_k(R) I_R(x; y) \qquad (13)$$

where $\alpha_k \geq 0$ and $\sum_k \alpha_k = 1$. The result is a regularizer with regions $\mathcal{K} = \cup_k \mathcal{R}^{(k)}$ and adjusted *a priori* weights $\alpha_k P_k(R)$ over the regions. The solution can therefore be found as before provided that $\{\alpha_k\}$ are known. When $\{\alpha_k\}$ are unknown, we set them competitively. In other words, we minimize the worst information rate across the available representations. This gives rise to the following regularization problem:

$$\max_{\alpha_k \geq 0, \sum \alpha_k = 1} \min_{Q(y|x)} J(Q; \lambda, \alpha) \qquad (14)$$

where $J(Q; \lambda, \alpha)$ is the overall objective that uses $J_c(Q; \mathcal{K}, \alpha)$ as the regularizer. The maximum is well-defined since the objective is concave in $\{\alpha_k\}$. This follows immediately as the objective is a minimum of a collection of linear functions $J(Q; \lambda, \alpha)$ (linear in $\{\alpha_k\}$).

At the optimum all $J_c(Q; \mathcal{R}^{(k)})$ for which $\alpha_k > 0$ have the same value (the same information rate). Other feature sets, those with $\alpha_k = 0$, do not contribute to the overall solution as their information rates are dominated by others.

## 7  Experiments

We first illustrate the performance of information regularization on two generated binary classification tasks in the plane. Here we can derive a region covering from the Euclidean metric as spheres of a certain radius centered at each data point. On the data set in Figure 2 inspired from [3] the method correctly propagates the labels to the clusters starting

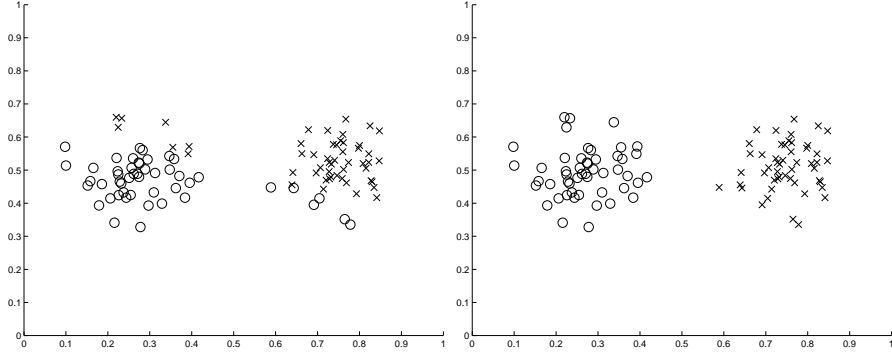

Figure 3: Ability of information regularization to correct the output of a prior classifier (left: before, right: after)

from a single labeled point in each class. In the example in Figure 3 we demonstrate that information regularization can be used as a post-processing to supervised classification and improve error rates by taking advantage of the topology of the space. All points are a priori labeled by a linear classifier that is non-optimal and places a decision boundary through the negative and positive clusters. Information regularization (on a Euclidean region covering defined as circles around each data point) is able to correct the mislabeling of the clusters. Next we test the algorithm on a web document classification task, the WebKB data set of [1]. The data consists of 1051 pages collected from the websites of four universities. This particular subset of WebKB is a binary classification task into 'course' and 'non-course' pages. 22% of the documents are positive ('course'). The dataset is interesting because apart from the documents contents we have information about the link structure of the documents. The two sources of information can illustrate the capability of information regularization of combining heterogeneous unlabeled representations.

Both 'text' and 'link' features used here are a bag-of-words representation of documents. To obtain 'link' features we collect text that appears under all links that link to that page from other pages, and produce its bag-of-words representation. We employ no stemming, or stop-word processing, but restrict the vocabulary to 2000 text words and 500 link words. The experimental setup consists of 100 random selections of 3 positive labeled, 9 negative labeled, and the rest unlabeled. The test set includes all unlabeled documents. We report a naïve Bayes baseline based on the model that features of different words are independent given the document class. The naïve Bayes algorithm can be run on text features, link features, or combine the two feature sets by assuming independence. We also quote the performance of the semi-supervised method obtained by combining naïve Bayes with the EM algorithm as in [9].

We measure the performance of the algorithms by the F-score equal to $2pr/(p+r)$, where $p$ and $r$ are the precision and recall. A high F-score indicates that the precision and recall are high and also close to each other. To compare algorithms independently of the probability threshold that decides between positive and negative samples, the results reported are the best F-scores for all possible settings of the threshold.

The key issue in applying information regularization is the derivation of a sound region covering $\mathcal{R}$. For document classification we obtained the best results by grouping all documents that share a certain word into the same region; thus each region is in fact a word, and there are as many regions as the size of the vocabulary. Regions are weighted equally, as well as the words belonging to the same region. The choice of $\lambda$ is also task dependent. Here cross-validation selected a optimal value $\lambda = 90$. When running information regu-

Table 1: Web page classification comparison between naïve Bayes and information regularization and semi-supervised naï ve Bayes+EM on text, link, and joint features

|      | naïve Bayes | inforeg | naïve Bayes+EM |
|------|-------------|---------|----------------|
| text | 82.85       | 85.10   | 93.69          |
| link | 65.64       | 82.85   | 67.18          |
| both | 83.33       | 86.15   | 91.01          |

larization with both text and link features we combined the coverings with a weight of $0.5$ rather than optimizing it in a min-max fashion.

All results are reported in Table 1. We observe that information regularization performs better than naïve Bayes on all types of features, that combining text and link features improves performance of the regularization method, and that on link features the method performs better than the semi-supervised naï ve Bayes+EM. Most likely the results do not reflect the full potential of information regularization due to the ad-hoc choice of regions based on the vocabulary used by naïve Bayes.

## 8 Discussion

The regularization principle introduced here provides a general information theoretic approach to exploiting unlabeled points. The solution implied by the principle is unique and can be found efficiently with distributed algorithms, performing complementary averages, on the graph induced by the regions. The propagation algorithms also extend to more structured settings. Our preliminary theoretical analysis concerning the number of possible labelings with bounded regularizer is suggestive but rather loose (tighter results can be found). The effect of the choice of the regions (sets of points that ought to be labeled the same) is critical in practice but not yet well-understood.

**References**

[1] A. Blum and T. Mitchell. Combining labeled and unlabeled data with co-training. In *Proceedings of the 1998 Conference on Computational Learning Theory*, 1998.

[2] X. Zhu, Z. Ghahramani, and J. Lafferty. Semi-supervised learning using gaussian fields and harmonic functions. In *Machine Learning: Proceedings of the Twentieth International Conference*, 2003.

[3] M. Szummer and T. Jaakkola. Partially labeled classification with markov random walks. In *Advances in Neural Information Processing Systems 14*, 2001.

[4] O. Chapelle, J. Weston, and B. Schoelkopf. Cluster kernels for semi-supervised learning. In *Advances in Neural Information Processing Systems 15*, 2002.

[5] M. Szummer and T. Jaakkola. Information regularization with partially labeled data. In *NIPS'2002*, volume 15, 2003.

[6] A. Corduneanu and T. Jaakkola. On information regularization. In *Proceedings of the 19th UAI*, 2003.

[7] T. M. Cover and J. A. Thomas. *Elements of Information Theory*. Wiley & Sons, New York, 1991.

[8] R. E. Blahut. Computation of channel capacity and rate distortion functions. In *IEEE Trans. Inform. Theory*, volume 18, pages 460–473, July 1972.

[9] K. Nigam, A.K. McCallum, S. Thrun, and T. Mitchell. Text classification from labeled and unlabeled documents using EM. *Machine Learning*, 39:103–134, 2000.

## Footnotes

[1]The result follows from $\sum_{i=0}^{k} \binom{n}{i} \leq \left( \frac{2n}{k} \right)^k$
